# Charting a Manifold

**Matthew Brand**
Mitsubishi Electric Research Labs
201 Broadway, Cambridge MA 02139 USA
www.merl.com/people/brand/

## Abstract

We construct a nonlinear mapping from a high-dimensional sample space to a low-dimensional vector space, effectively recovering a Cartesian coordinate system for the manifold from which the data is sampled. The mapping preserves local geometric relations in the manifold and is pseudo-invertible. We show how to estimate the intrinsic dimensionality of the manifold from samples, decompose the sample data into locally linear low-dimensional patches, merge these patches into a single low-dimensional coordinate system, and compute forward and reverse mappings between the sample and coordinate spaces. The objective functions are convex and their solutions are given in closed form.

## 1 Nonlinear dimensionality reduction (NLDR) by charting

Charting is the problem of assigning a low-dimensional coordinate system to data points in a high-dimensional sample space. It is presumed that the data lies on or near a low-dimensional manifold embedded in the sample space, and that there exists a 1-to-1 smooth nonlinear transform between the manifold and a low-dimensional vector space. The data-modeler's goal is to estimate smooth continuous mappings between the sample and coordinate spaces. Often this analysis will shed light on the intrinsic variables of the data-generating phenomenon, for example, revealing perceptual or configuration spaces.

Our goal is to find a mapping—expressed as a kernel-based mixture of linear projections—that minimizes information loss about the density and relative locations of sample points. This constraint is expressed in a posterior that combines a standard gaussian mixture model (GMM) likelihood function with a prior that penalizes uncertainty due to inconsistent projections in the mixture. Section 3 develops a special case where this posterior is unimodal and maximizable in closed form, yielding a GMM whose covariances reveal a patchwork of overlapping locally linear subspaces that cover the manifold. Section 4 shows that for this (or *any*) GMM and a choice of reduced dimension $d$, there is a unique, closed-form solution for a minimally distorting merger of the subspaces into a $d$-dimensional coordinate space, as well as an reverse mapping defining the surface of the manifold in the sample space. The intrinsic dimensionality $d$ of the data manifold can be estimated from the growth process of point-to-point distances. In analogy to differential geometry, we call the subspaces "charts" and their merger the "connection." Section 5 considers example problems where these methods are used to untie knots, unroll and untwist sheets, and visualize video data.

### 1.1 Background

Topology-neutral NLDR algorithms can be divided into those that compute mappings, and

those that directly compute low-dimensional embeddings. The field has its roots in mapping algorithms: DeMers and Cottrell [3] proposed using auto-encoding neural networks with a hidden layer "bottleneck," effectively casting dimensionality reduction as a compression problem. Hastie defined principal curves [5] as nonparametric 1D curves that pass through the center of "nearby" data points. A rich literature has grown up around properly regularizing this approach and extending it to surfaces. Smola and colleagues [10] analyzed the NLDR problem in the broader framework of regularized quantization methods.

More recent advances aim for embeddings: Gomes and Mojsilovic [4] treat manifold completion as an anisotropic diffusion problem, iteratively expanding points until they connect to their neighbors. The ISOMAP algorithm [12] represents remote distances as sums of a trusted set of distances between immediate neighbors, then uses multidimensional scaling to compute a low-dimensional embedding that minimally distorts all distances. The locally linear embedding algorithm (LLE) [9] represents each point as a weighted combination of a trusted set of nearest neighbors, then computes a minimally distorting low-dimensional barycentric embedding. They have complementary strengths: ISOMAP handles holes well but can fail if the data hull is nonconvex [12]; and vice versa for LLE [9]. Both offer embeddings without mappings. It has been noted that trusted-set methods are vulnerable to noise because they consider the subset of point-to-point relationships that has the lowest signal-to-noise ratio; small changes to the trusted set can induce large changes in the set of constraints on the embedding, making solutions unstable [1].

In a return to mapping, Roweis and colleagues [8] proposed global coordination—learning a mixture of locally linear projections from sample to coordinate space. They constructed a posterior that penalizes distortions in the mapping, and gave a expectation-maximization (EM) training rule. Innovative use of variational methods highlighted the difficulty of even hill-climbing their multimodal posterior. Like [2, 7, 6, 8], the method we develop below is a decomposition of the manifold into locally linear neighborhoods. It bears closest relation to global coordination [8], although by a different construction of the problem, we avoid hill-climbing a spiky posterior and instead develop a closed-form solution.

## 2    Estimating locally linear scale and intrinsic dimensionality

We begin with matrix of sample points $\mathbf{Y} \doteq [\mathbf{y}_1, \cdots, \mathbf{y}_N], \mathbf{y}_n \in \mathbb{R}^D$ populating a $D$-dimensional sample space, and a conjecture that these points are samples from a manifold $\mathcal{M}$ of intrinsic dimensionality $d < D$. We seek a mapping onto a vector space $G(\mathbf{Y}) \to \mathbf{X} \doteq [\mathbf{x}_1, \cdots, \mathbf{x}_N], \mathbf{x}_n \in \mathbb{R}^d$ and 1-to-1 reverse mapping $G^{-1}(\mathbf{X}) \to \mathbf{Y}$ such that local relations between nearby points are preserved (this will be formalized below). The map $G$ should be non-catastrophic, that is, without folds: Parallel lines on the manifold in $\mathbb{R}^D$ should map to continuous smooth non-intersecting curves in $\mathbb{R}^d$. This guarantees that linear operations on $\mathbf{X}$ such as interpolation will have reasonable analogues on $\mathbf{Y}$.

Smoothness means that at some scale $r$ the mapping from a neighborhood on $\mathcal{M}$ to $\mathbb{R}^d$ is effectively linear. Consider a ball of radius $r$ centered on a data point and containing $n(r)$ data points. The count $n(r)$ grows as $r^d$, but *only* at the locally linear scale; the grow rate is inflated by isotropic noise at smaller scales and by embedding curvature at larger scales. To estimate $r$, we look at how the $r$-ball grows as points are added to it, tracking $c(r) \doteq \frac{d}{d \log n(r)} \log r$. At noise scales, $c(r) \approx 1/D < 1/d$, because noise has distributed points in all directions with equal probability. At the scale at which curvature becomes significant, $c(r) < 1/d$, because the manifold is no longer perpendicular to the surface of the ball, so the ball does not have to grow as fast to accommodate new points. At the locally linear scale, the process *peaks* at $c(r) = 1/d$, because points are distributed only in the directions of the manifold's local tangent space. The maximum of $c(r)$ therefore gives an estimate of both the scale and the local dimensionality of the manifold (see figure 1), provided that the ball hasn't expanded to a manifold boundary—boundaries have lower dimension than

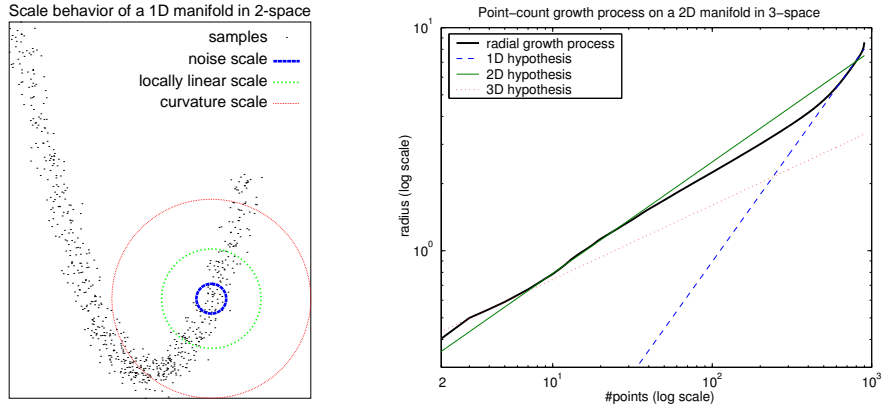

Figure 1: Point growth processes. LEFT: At the locally linear scale, the number of points in an $r$-ball grows as $r^d$; at noise and curvature scales it grows faster. RIGHT: Using the point-count growth process to find the intrinsic dimensionality of a 2D manifold nonlinearly embedded in 3-space (see figure 2). Lines of slope $1/3$, $1/2$, and 1 are fitted to sections of the $\log r / \log n_r$ curve. For neighborhoods of radius $r \approx 1$ with roughly $n \approx 10$ points, the slope peaks at $1/2$ indicating a dimensionality of $d = 2$. Below that, the data appears 3D because it is dominated by noise (except for $n \leq D$ points); above, the data appears >2D because of manifold curvature. As the $r$-ball expands to cover the entire data-set the dimensionality appears to drop to 1 as the process begins to track the 1D edges of the 2D sheet.

the manifold. For low-dimensional manifolds such as sheets, the boundary submanifolds (edges and corners) are very small relative to the full manifold, so the boundary effect is typically limited to a small rise in $c(r)$ as $r$ approaches the scale of the entire data set. In practice, our code simply expands an $r$-ball at every point and looks for the first peak in $c(r)$, averaged over many nearby $r$-balls. One can estimate $d$ and $r$ globally or per-point.

## 3 Charting the data

In the charting step we find a soft partitioning of the data into locally linear low-dimensional neighborhoods, as a prelude to computing the connection that gives the global low-dimensional embedding. To minimize information loss in the connection, we require that the data points project into a subspace associated with each neighborhood with (1) minimal loss of local variance and (2) maximal agreement of the projections of nearby points into nearby neighborhoods. Criterion (1) is served by maximizing the likelihood function of a Gaussian mixture model (GMM) density fitted to the data:

$$p(\mathbf{y}_i|\mu,\Sigma) \doteq \sum_j p(\mathbf{y}_i|\mu_j,\Sigma_j)\, p_j = \sum_j \mathcal{N}(\mathbf{y}_i;\mu_j,\Sigma_j)\, p_j \quad . \tag{1}$$

Each gaussian component defines a local neighborhood centered around $\mu_j$ with axes defined by the eigenvectors of $\Sigma_j$. The amount of data variance along each axis is indicated by the eigenvalues of $\Sigma_j$; if the data manifold is locally linear in the vicinity of the $\mu_j$, all but the $d$ dominant eigenvalues will be near-zero, implying that the associated eigenvectors constitute the optimal variance-preserving local coordinate system. To some degree likelihood maximization will naturally realize this property: It requires that the GMM components shrink in volume to fit the data as tightly as possible, which is best achieved by positioning the components so that they "pancake" onto locally flat collections of data-points. However, this state of affairs is easily violated by degenerate (zero-variance) GMM components or components fitted to overly small enough locales where the data density off the manifold is comparable to density on the manifold (e.g., at the noise scale). Consequently a prior is needed.

Criterion (2) implies that neighboring partitions should have dominant axes that span similar subspaces, since disagreement (large subspace angles) would lead to inconsistent projections of a point and therefore uncertainty about its location in a low-dimensional coordinate space. The principal insight is that criterion (2) is exactly the cost of coding the location of a point in one neighborhood when it is generated by another neighborhood—the cross-entropy between the gaussian models defining the two neighborhoods:

$$
\begin{aligned}
D(\mathcal{N}_1 \| \mathcal{N}_2) &= \int d\mathbf{y}\, \mathcal{N}(\mathbf{y};\mu_1,\Sigma_1) \log \frac{\mathcal{N}(\mathbf{y};\mu_1,\Sigma_1)}{\mathcal{N}(\mathbf{y};\mu_2,\Sigma_2)} \\
&= (\log|\Sigma_1^{-1}\Sigma_2| + \text{trace}(\Sigma_2^{-1}\Sigma_1) + (\mu_2-\mu_1)^\top \Sigma_2^{-1}(\mu_2-\mu_1) - D)/2. \quad (2)
\end{aligned}
$$

Roughly speaking, the terms in (2) measure differences in size, orientation, and position, respectively, of two coordinate frames located at the means $\mu_1, \mu_2$ with axes specified by the eigenvectors of $\Sigma_1, \Sigma_2$. All three terms decline to zero as the overlap between the two frames is maximized. To maximize consistency between adjacent neighborhoods, we form the prior $p(\mu,\Sigma) \doteq \exp[-\sum_{i \neq j} m_i(\mu_j) D(\mathcal{N}_i \| \mathcal{N}_j)]$, where $m_i(\mu_j)$ is a measure of co-locality.

Unlike global coordination [8], we are not asking that the dominant axes in neighboring charts are aligned—only that they span nearly the same subspace. This is a much easier objective to satisfy, and it contains a useful special case where the posterior $p(\mu,\Sigma|\mathbf{Y}) \propto \sum_i p(\mathbf{y}_i|\mu,\Sigma)p(\mu,\Sigma)$ is unimodal and can be maximized in closed form: Let us associate a gaussian neighborhood with each data-point, setting $\mu_i = \mathbf{y}_i$; take all neighborhoods to be *a priori* equally probable, setting $p_i = 1/N$; and let the co-locality measure be determined from some local kernel. For example, in this paper we use $m_i(\mu_j) \propto \mathcal{N}(\mu_j;\mu_i,\sigma^2)$, with the scale parameter $\sigma$ specifying the expected size of a neighborhood on the manifold in sample space. A reasonable choice is $\sigma = r/2$, so that $2\text{erf}(2) > 99.5\%$ of the density of $m_i(\mu_j)$ is contained in the area around $\mathbf{y}_i$ where the manifold is expected to be locally linear. With uniform $p_i$ and $\mu_i$, $m_i(\mu_j)$ and fixed, the MAP estimates of the GMM covariances are

$$
\Sigma_i = \left( \sum_j m_i(\mu_j)\left( (\mathbf{y}_j - \mu_i)(\mathbf{y}_j - \mu_i)^\top + (\mu_j - \mu_i)(\mu_j - \mu_i)^\top + \Sigma_j \right) \right) \Big/ \sum_j m_i(\mu_j) \quad (3)
$$

Note that each covariance $\Sigma_i$ is dependent on all other $\Sigma_j$. The MAP estimators for all covariances can be arranged into a set of fully constrained linear equations and solved exactly for their mutually optimal values. This key step brings nonlocal information about the manifold's shape into the local description of each neighborhood, ensuring that adjoining neighborhoods have similar covariances and small angles between their respective subspaces. Even if a local subset of data points are dense in a direction *perpendicular* to the manifold, the prior encourages the local chart to orient *parallel* to the manifold as part of a globally optimal solution, protecting against a pathology noted in [8]. Equation (3) is easily adapted to give a reduced number of charts and/or charts centered on local centroids.

## 4 Connecting the charts

We now build a connection for set of charts specified as an arbitrary nondegenerate GMM. A GMM gives a soft partitioning of the dataset into neighborhoods of mean $\mu_k$ and covariance $\Sigma_k$. The optimal variance-preserving low-dimensional coordinate system for each neighborhood derives from its weighted principal component analysis, which is exactly specified by the eigenvectors of its covariance matrix: Eigendecompose $\mathbf{V}_k \Lambda_k \mathbf{V}_k^\top \leftarrow \Sigma_k$ with eigenvalues in descending order on the diagonal of $\Lambda_k$ and let $\mathbf{W}_k \doteq [\mathbf{I}_d, \mathbf{0}]\mathbf{V}_k^\top$ be the operator projecting points into the $k^{th}$ local chart, such that local chart coordinate $\mathbf{u}_{ki} \doteq \mathbf{W}_k(\mathbf{y}_i - \mu_k)$ and $\mathbf{U}_k \doteq [\mathbf{u}_{k1}, \cdots, \mathbf{u}_{kN}]$ holds the local coordinates of all points.

Our goal is to sew together all charts into a globally consistent low-dimensional coordinate system. For each chart there will be a low-dimensional affine transform $\mathbf{G}_k \in \mathbb{R}^{(d+1)\times d}$

that projects $\mathbf{U}_k$ into the global coordinate space. Summing over all charts, the weighted average of the projections of point $\mathbf{y}_i$ into the low-dimensional vector space is

$$\widehat{\mathbf{x}|\mathbf{y}} \doteq \sum_j \mathbf{G}_j \begin{bmatrix} \mathbf{W}_j(\mathbf{y}-\mu_j) \\ 1 \end{bmatrix} p_{j|\mathbf{y}}(\mathbf{y}) \qquad \Rightarrow \qquad \widehat{\mathbf{x}_i|\mathbf{y}_i} \doteq \sum_j \mathbf{G}_j \begin{bmatrix} \mathbf{u}_{ji} \\ 1 \end{bmatrix} p_{j|\mathbf{y}}(\mathbf{y}_i), \quad (4)$$

where $p_{k|\mathbf{y}}(\mathbf{y}) \propto p_k \mathcal{N}(\mathbf{y};\mu_k,\Sigma_k)$, $\sum_k p_{k|\mathbf{y}}(\mathbf{y}) = 1$ is the probability that chart $k$ generates point $\mathbf{y}$. As pointed out in [8], if a point has nonzero probabilities in two charts, then there should be affine transforms of those two charts that map the point to the same place in a global coordinate space. We set this up as a weighted least-squares problem:

$$\mathbf{G} \doteq [\mathbf{G}_1, \cdots, \mathbf{G}_K] = \arg\min_{\mathbf{G}_k, \mathbf{G}_j} \sum_i p_{k|\mathbf{y}}(\mathbf{y}_i) p_{j|\mathbf{y}}(\mathbf{y}_i) \left\| \mathbf{G}_k \begin{bmatrix} \mathbf{u}_{ki} \\ 1 \end{bmatrix} - \mathbf{G}_j \begin{bmatrix} \mathbf{u}_{ji} \\ 1 \end{bmatrix} \right\|_F^2 . \quad (5)$$

Equation (5) generates a homogeneous set of equations that determines a solution up to an affine transform of $\mathbf{G}$. There are two solution methods. First, let us temporarily anchor one neighborhood at the origin to fix this indeterminacy. This adds the constraint $\mathbf{G}_1 = [\mathbf{I}, \mathbf{0}]^\top$.

To solve, define indicator matrix $\mathbf{F}_k \doteq [\mathbf{0}, \cdots, \mathbf{0}, \mathbf{I}, \mathbf{0}, \cdots, \mathbf{0}]^\top$ with the identity matrix occupying the $k^{th}$ block, such that $\mathbf{G}_k = \mathbf{G}\mathbf{F}_k$. Let the diagonal of $\mathbf{P}_k \doteq \mathrm{diag}([\mathrm{p}_{k|\mathbf{y}}(\mathbf{y}_1), \cdots, \mathrm{p}_{k|\mathbf{y}}(\mathbf{y}_N)])$ record the per-point posteriors of chart $k$. The squared error of the connection is then a sum of of all patch-to-anchor and patch-to-patch inconsistencies:

$$\mathcal{E} \doteq \sum_k \left[ \left\| (\mathbf{G}\underline{\mathbf{U}}_k - \begin{bmatrix} \mathbf{U}_1 \\ \mathbf{0} \end{bmatrix}) \mathbf{P}_k \mathbf{P}_1 \right\|_F^2 + \sum_{j \neq k} \left\| (\mathbf{G}\underline{\mathbf{U}}_j - \mathbf{G}\underline{\mathbf{U}}_k) \mathbf{P}_j \mathbf{P}_k \right\|_F^2 \right]; \quad \underline{\mathbf{U}}_k \doteq \mathbf{F}_k \begin{bmatrix} \mathbf{U}_k \\ 1 \end{bmatrix} . \quad (6)$$

Setting $\mathrm{d}\mathcal{E}/\mathrm{d}\mathbf{G} = 0$ and solving to minimize convex $\mathcal{E}$ gives

$$\mathbf{G}^\top = \left( \sum_k \underline{\mathbf{U}}_k \mathbf{P}_k^2 \left( \sum_{j \neq k} \mathbf{P}_j^2 \right) \underline{\mathbf{U}}_k^\top - \sum_{j \neq k} \underline{\mathbf{U}}_k \mathbf{P}_k^2 \mathbf{P}_j^2 \underline{\mathbf{U}}_j^\top \right)^{-1} \left( \sum_k \underline{\mathbf{U}}_k \mathbf{P}_k^2 \mathbf{P}_1^2 \begin{bmatrix} \mathbf{U}_1 \\ \mathbf{0} \end{bmatrix}^\top \right). \quad (7)$$

We now remove the dependence on a reference neighborhood $\mathbf{G}_1$ by rewriting equation 5,

$$\mathbf{G} = \arg\min_{\mathbf{G}} \left( \textstyle\sum_{j \neq k} \| (\mathbf{G}\underline{\mathbf{U}}_j - \mathbf{G}\underline{\mathbf{U}}_k) \mathbf{P}_j \mathbf{P}_k \|_F^2 = \| \mathbf{G}\mathbf{Q} \|_F^2 = \mathrm{trace}(\mathbf{G}\mathbf{Q}\mathbf{Q}^\top \mathbf{G}^\top) \right), \quad (8)$$

where $\mathbf{Q} \doteq \sum_{j \neq k} \left( (\underline{\mathbf{U}}_j - \underline{\mathbf{U}}_k) \mathbf{P}_j \mathbf{P}_k \right)$. If we require that $\mathbf{G}\mathbf{G}^\top = \mathbf{I}$ to prevent degenerate solutions, then equation (8) is solved (up to rotation in coordinate space) by setting $\mathbf{G}^\top$ to the eigenvectors associated with the smallest eigenvalues of $\mathbf{Q}\mathbf{Q}^\top$. The eigenvectors can be computed efficiently without explicitly forming $\mathbf{Q}\mathbf{Q}^\top$; other numerical efficiencies obtain by zeroing any vanishingly small probabilities in each $\mathbf{P}_k$, yielding a sparse eigenproblem.

A more interesting strategy is to numerically condition the problem by calculating the trailing eigenvectors of $\mathbf{Q}\mathbf{Q}^\top + \mathbf{1}$. It can be shown that this maximizes the posterior $p(\mathbf{G}|\mathbf{Q}) \propto p(\mathbf{Q}|\mathbf{G})p(\mathbf{G}) \propto e^{-\|\mathbf{G}\mathbf{Q}\|_F^2} e^{-\|\mathbf{G}\mathbf{1}\|}$, where the prior $p(\mathbf{G})$ favors a mapping $\mathbf{G}$ whose unit-norm rows are also zero-mean. This maximizes variance in each row of $\mathbf{G}$ and thereby spreads the projected points broadly and evenly over coordinate space.

The solutions for MAP charts (equation (5)) and connection (equation (8)) can be applied to any well-fitted mixture of gaussians/factors[1]/PCAs density model; thus large eigenproblems can be avoided by connecting just a small number of charts that cover the data.

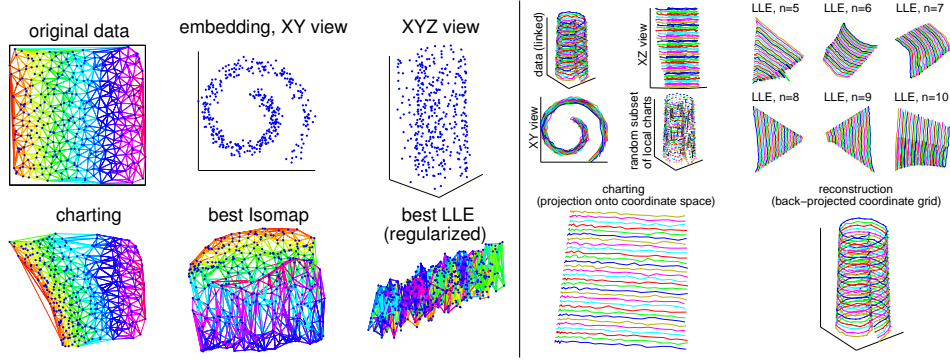

Figure 2: The twisted curl problem. LEFT: Comparison of charting, ISOMAP, & LLE. 400 points are randomly sampled from the manifold with noise. Charting is the only method that recovers the original space without catastrophes (folding), albeit with some shear. RIGHT: The manifold is regularly sampled (with noise) to illustrate the forward and backward projections. Samples are shown linked into lines to help visualize the manifold structure. Coordinate axes of a random selection of charts are shown as bold lines. Connecting subsets of charts such as this will also give good mappings. The upper right quadrant shows various LLE results. At bottom we show the charting solution and the reconstructed (back-projected) manifold, which smooths out the noise.

Once the connection is solved, equation (4) gives the forward projection of any point $\mathbf{y}$ down into coordinate space. There are several numerically distinct candidates for the back-projection: posterior mean, mode, or exact inverse. In general, there may not be a unique posterior mode and the exact inverse is not solvable in closed form (this is also true of [8]). Note that chart-wise projection defines a complementary density in coordinate space

$$p_{\mathbf{x}|k}(\mathbf{x}) = \mathcal{N}(\mathbf{x}; \mathbf{G}_k \begin{bmatrix} \mathbf{0} \\ 1 \end{bmatrix}, \mathbf{G}_k \begin{bmatrix} [\mathbf{I}_d, \mathbf{0}]\Lambda_k[\mathbf{I}_d, \mathbf{0}]^\top & \mathbf{0} \\ \mathbf{0} & 0 \end{bmatrix} \mathbf{G}_k^\top). \tag{9}$$

Let $p(\mathbf{y}|\mathbf{x}, k)$, used to map $\mathbf{x}$ into subspace $k$ on the surface of the manifold, be a Dirac delta function whose mean is a linear function of $\mathbf{x}$. Then the posterior mean back-projection is obtained by integrating out uncertainty over which chart generates $\mathbf{x}$:

$$\widehat{\mathbf{y}|\mathbf{x}} = \sum_k p_{k|\mathbf{x}}(\mathbf{x}) \left( \mu_k + \mathbf{W}_k^\top \left( \mathbf{G}_k \begin{bmatrix} \mathbf{I} \\ 0 \end{bmatrix} \right)^+ \left( \mathbf{x} - \mathbf{G}_k \begin{bmatrix} \mathbf{0} \\ 1 \end{bmatrix} \right) \right), \tag{10}$$

where $(\cdot)^+$ denotes pseudo-inverse. In general, a back-projecting map should *not* reconstruct the original points. Instead, equation (10) generates a surface that passes through the weighted average of the $\mu_i$ of all the neighborhoods in which $\mathbf{y}_i$ has nonzero probability, much like a principal curve passes through the center of each local group of points.

## 5 Experiments

**Synthetic examples:** 400 2D points were randomly sampled from a 2D square and embedded in 3D via a curl and twist, then contaminated with gaussian noise. Even if noiselessly sampled, this manifold cannot be "unrolled" without distortion. In addition, the outer curl is sampled much less densely than the inner curl. With an order of magnitude fewer points, higher noise levels, no possibility of an isometric mapping, and uneven sampling, this is arguably a much more challenging problem than the "swiss roll" and "s-curve" problems featured in [12, 9, 8, 1]. Figure 2LEFT contrasts the (unique) output of charting and the best outputs obtained from ISOMAP and LLE (considering all neighborhood sizes between 2 and 20 points). ISOMAP and LLE show catastrophic folding; we had to change LLE's

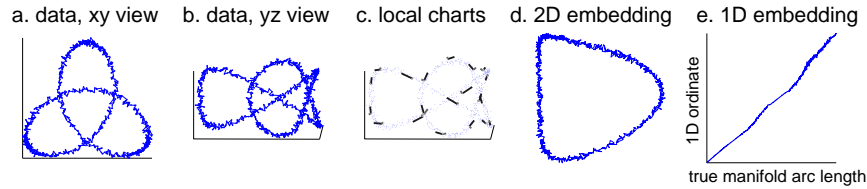

Figure 3: Untying a trefoil knot (⟨knot symbol⟩) by charting. 900 noisy samples from a 3D-embedded 1D manifold are shown as connected dots in front (a) and side (b) views. A subset of charts is shown in (c). Solving for the 2D connection gives the "unknot" in (d). After removing some points to cut the knot, charting gives a 1D embedding which we plot against true manifold arc length in (e); monotonicity (modulo noise) indicates correctness.

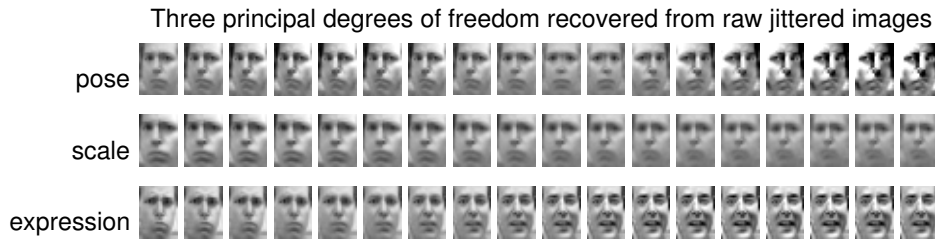

Figure 4: Modeling the manifold of facial images from raw video. Each row contains images synthesized by back-projecting an axis-parallel straight line in coordinate space onto the manifold in image space. Blurry images correspond to points on the manifold whose neighborhoods contain few if any nearby data points.

regularization in order to coax out nondegenerate (>1D) solutions. Although charting is not designed for isometry, after affine transform the forward-projected points disagree with the original points with an RMS error of only 1.0429, lower than the best LLE (3.1423) or best ISOMAP (1.1424, not shown). Figure 2RIGHT shows the same problem where points are sampled regularly from a grid, with noise added before and after embedding. Figure 3 shows a similar treatment of a 1D line that was threaded into a 3D trefoil knot, contaminated with gaussian noise, and then "untied" via charting.

**Video:** We obtained a 1965-frame video sequence (courtesy S. Roweis and B. Frey) of $20 \times 28$-pixel images in which B.F. strikes a variety of poses and expressions. The video is heavily contaminated with synthetic camera jitters. We used raw images, though image processing could have removed this and other uninteresting sources of variation. We took a 500-frame subsequence and left-right mirrored it to obtain 1000 points in $20 \times 28 = 560$D image space. The point-growth process peaked just above $d = 3$ dimensions. We solved for 25 charts, each centered on a random point, and a 3D connection. The recovered degrees of freedom—recognizable as pose, scale, and expression—are visualized in figure 4.

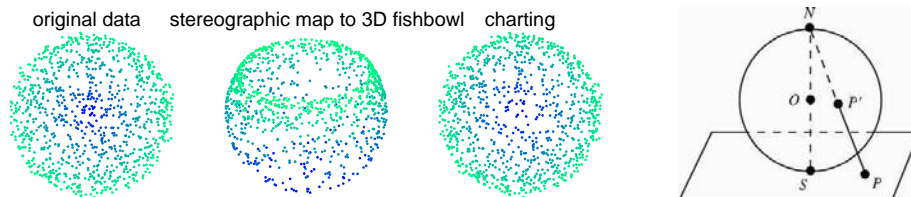

Figure 5: Flattening a fishbowl. From the left: Original $2000 \times 2$D points; their stereographic mapping to a 3D fishbowl; its 2D embedding recovered using 500 charts; and the stereographic map. Fewer charts lead to isometric mappings that fold the bowl (not shown).

**Conformality:** Some manifolds can be flattened conformally (preserving local angles) but not isometrically. Figure 5 shows that if the data is finely charted, the connection behaves more conformally than isometrically. This problem was suggested by J. Tenenbaum.

# 6   Discussion

Charting breaks kernel-based NLDR into two subproblems: (1) Finding a set of data-covering locally linear neighborhoods ("charts") such that adjoining neighborhoods span maximally similar subspaces, and (2) computing a minimal-distortion merger ("connection") of all charts. The solution to (1) is optimal w.r.t. the estimated scale of local linearity $r$; the solution to (2) is optimal w.r.t. the solution to (1) and the desired dimensionality $d$. Both problems have Bayesian settings. By offloading the nonlinearity onto the kernels, we obtain least-squares problems and closed form solutions. This scheme is also attractive because large eigenproblems can be avoided by using a reduced set of charts.

The dependence on $r$, like trusted-set methods, is a potential source of solution instability. In practice the point-growth estimate seems fairly robust to data perturbations (to be expected if the data density changes slowly over a manifold of integral Hausdorff dimension), while the use of a soft neighborhood partitioning appears to make charting solutions reasonably stable to variations in $r$. Eigenvalue stability analyses may prove useful here. Ultimately, we would prefer to integrate $r$ out. In contrast, use of $d$ appears to be a virtue: Unlike other eigenvector-based methods, the best $d$-dimensional embedding is not merely a linear projection of the best $d+1$-dimensional embedding; a unique distortion is found for each value of $d$ that maximizes the information content of its embedding.

Why does charting performs well on datasets where the signal-to-noise ratio confounds recent state-of-the-art methods? Two reasons may be adduced: (1) Nonlocal information is used to construct both the system of local charts and their global connection. (2) The mapping only preserves the component of local point-to-point distances that project *onto* the manifold; relationships perpendicular to the manifold are discarded. Thus charting uses global shape information to suppress noise in the constraints that determine the mapping.

### Acknowledgments

Thanks to J. Buhmann, S. Makar, S. Roweis, J. Tenenbaum, and anonymous reviewers for insightful comments and suggested "challenge" problems.

## Footnotes

[1]We thank reviewers for calling our attention to Teh & Roweis ([11]—in this volume), which shows how to connect a set of *given* local dimensionality reducers in a generalized eigenvalue problem that is related to equation (8).

# References

[1]   M. Balasubramanian and E. L. Schwartz. The IsoMap algorithm and topological stability. *Science*, 295(5552):7, January 2002.

[2]   C. Bregler and S. Omohundro. Nonlinear image interpolation using manifold learning. In *NIPS–7*, 1995.

[3]   D. DeMers and G. Cottrell. Nonlinear dimensionality reduction. In *NIPS–5*, 1993.

[4]   J. Gomes and A. Mojsilovic. A variational approach to recovering a manifold from sample points. In *ECCV*, 2002.

[5]   T. Hastie and W. Stuetzle. Principal curves. *J. Am. Statistical Assoc*, 84(406):502–516, 1989.

[6]   G. Hinton, P. Dayan, and M. Revow. Modeling the manifolds of handwritten digits. *IEEE Trans. Neural Networks*, 8, 1997.

[7]   N. Kambhatla and T. Leen. Dimensionality reduction by local principal component analysis. *Neural Computation*, 9, 1997.

[8]   S. Roweis, L. Saul, and G. Hinton. Global coordination of linear models. In *NIPS–13*, 2002.

[9]   S. T. Roweis and L. K. Saul. Nonlinear dimensionality reduction by locally linear embedding. *Science*, 290:2323–2326, December 22 2000.

[10]   A. Smola, S. Mika, B. Schölkopf, and R. Williamson. Regularized principal manifolds. *Machine Learning*, 1999.

[11]   Y. W. Teh and S. T. Roweis. Automatic alignment of hidden representations. In *NIPS–15*, 2003.

[12]   J. B. Tenenbaum, V. de Silva, and J. C. Langford. A global geometric framework for nonlinear dimensionality reduction. *Science*, 290:2319–2323, December 22 2000.
